# Generalization Error and The Expected Network Complexity

**Chuanyi Ji**
Dept. of Elec., Compt. and Syst Engr.
Rensselaer Polytechnic Institute
Troy, NY 12180-3590
chuanyi@ecse.rpi.edu

## Abstract

For two layer networks with $n$ sigmoidal hidden units, the generalization error is shown to be bounded by

$$O(\mathbf{E}\frac{1}{K}) + O(\frac{(\mathbf{E}K)d}{N}logN),$$

where $d$ and $N$ are the input dimension and the number of training samples, respectively. $\mathbf{E}$ represents the expectation on random number $K$ of hidden units $(1 \leq K \leq n)$. The probability $\Pr(K = k)$ $(1 \leq k \leq n)$ is determined by a prior distribution of weights, which corresponds to a Gibbs distribution of a regularizer. This relationship makes it possible to characterize explicitly how a regularization term affects bias/variance of networks. The bound can be obtained analytically for a large class of commonly used priors. It can also be applied to estimate the expected network complexity $\mathbf{E}K$ in practice. The result provides a quantitative explanation on how large networks can generalize well.

## 1  Introduction

Regularization (or weight-decay) methods are widely used in supervised learning by adding a regularization term to an energy function. Although it is well known that such a regularization term effectively reduces network complexity by introducing more bias and less variance[4] to the networks, it is not clear whether and how the information given by a regularization term can be used alone to characterize the effective network complexity and how the estimated effective network complexity relates to the generalization error. This research attempts to provide answers to these questions for two layer feedforward networks with sigmoidal hidden units.

Specifically, the effective network complexity is characterized by the expected num­ber of hidden units determined by a Gibbs distribution corresponding to a regular­ization term. The generalization error can then be bounded by the expected network complexity, and thus be tighter than the original bound given by Barron[2]. The new bound shows explicitly, through a bigger approximation error and a smaller estimation error, how a regularization term introduces more bias and less variance to the networks. It therefore provides a quantitative explanation on how a network larger than necessary can also generalize well under certain conditions, which can not be explained by the existing learning theory[9].

For a class of commonly-used regularizers, the expected network complexity can be obtained in a closed form. It is then used to estimate the expected network complexity for Gaussion mixture model[6].

## 2    Background and Previous Results

A relationship has been developed by Barron[2] between generalization error and network complexity for two layer networks used for function approximation. We will briefly describe this result in this section and give our extension subsequently.

Consider a class of two layer networks of fixed architecture with $n$ sigmoidal hidden units and one (linear) output unit. Let $f_n(x; w) = \sum_{l=1}^{n} w_l^{(2)} g_l(w_l^{(1)T} x)$ be a *network function*, where $w \in \Theta_n$ is the network weight vector comprising both $w_l^{(2)}$ and $w_l^{(1)}$ for $1 \leq l \leq n$. $w_l^{(1)}$ and $w_l^{(2)}$ are the incoming weights to the $l$-th hidden unit and the weight from the $l$-th hidden unit to the output, respectively. $\Theta_n \subseteq R^{n(d+1)}$ is the weight space for $n$ hidden units (and input dimension $d$). Each sigmoid unit $g_l(z)$ is assumed to be of tanh type: $g_l(z) \to \pm 1$ as $z \to \pm\infty$ for $1 \leq l \leq n$ [1]. The input is $x \in D \subseteq R^d$. Without loss of generality, $D$ is assumed to be a unit hypercube in $R^d$, i.e., all the components of $x$ are in $[-1, 1]$.

Let $f(x)$ be a target function defined in the same domain $D$ and satisfy some smoothness conditions [2]. Consider $N$ training samples independently drawn from some distribution $\mu(x)$: $(x_1, f(x_1)), ..., (x_N, f(x_N))$. Define an energy function $e$, where $e = \epsilon_1 + \lambda \frac{L_{n,N}(w)}{N}$. $L_{n,N}(w)$ is a regularization term as a function of $w$ for a fixed $n$. $\lambda$ is a constant. $c_1$ is a quadratic error function on $N$ training samples: $e_1 = \frac{1}{N} \sum_{i=1}^{N} (f_n(x_i; w) - f(x_i))^2$. Let $f_{n,N}(x; \hat{w})$ be the (optimal) network function such that $\hat{w}$ minimizes the energy function $e$: $\hat{w} = \arg \min_{w \in \Theta_n} e$. The gen­eralization error $E_g$ is defined to be the squared $L^2$ norm $E_g = \mathbf{E}\| f - f_{n,N} \|^2 = \mathbf{E}\int_D (f(x) - f_{n,N}(x; \hat{w}))^2 d\mu(x)$, where $\mathbf{E}$ is the expectation over all training sets of size $N$ drawn from the same distribution. Thus, the generalization error measures the mean squared distance between the unknown function and the best network function that can be obtained for training sets of size $N$. The generalization error

$E_g$ is shown[2] to be bounded as

$$E_g \leq O(R_{n,N}), \tag{1}$$

where $R_{n,N}$, called the index of resolvability [2], can be expressed as

$$R_{n,N} = \min_{w \in \Theta_n} \{ \| f - \hat{f}_n \|^2 + \frac{L_{n,N}(w)}{N} \}, \tag{2}$$

where $\hat{f}_n$ is the clipped $f_n(x; w)$ (see [2]). The index of resolvability can be further bounded as $R_{n,N} \leq O(\frac{1}{n}) + O(\frac{nd}{N} log N)$. Therefore, the generalization error is bounded as

$$E_g \leq O(\frac{1}{n}) + O(\frac{nd}{N} log N), \tag{3}$$

where $O(\frac{1}{n})$ and $O(\frac{nd}{N} log N)$ are the bounds for approximation error (bias) and estiamtion error (variance), respectively.

In addition, the bound for $E_g$ can be minimized if an additional regularization term $L_N(n)$ is used in the energy function to minimize the number of hidden units, i.e., $E_g \leq O(\sqrt{\frac{N}{d log N}})$.

## 3   Open Questions and Motivations

Two open questions, which can not be answered by the previous result, are of the primary interest of this work.

1) How do large networks generalize?

The large networks refer to those with a ratio $\frac{W}{N}$ to be somewhat big, where $W$ and $N$ are the total number of independently modifiable weights ($W \approx nd$, for $n$ large) and the number of training samples, respectively. Networks trained with regularization terms may fall into this category. Such large networks are found to be able to generalize well sometimes. However, when $\frac{nd}{N}$ is big, the bound in Equation (3) is too loose to bound the actual generalization error meaningfully. Therefore, for the large networks, the total number of hidden units $n$ may no longer be a good estimate for network complexity. Efforts have been made to develop measures on effective network complexity both analytically and empirically[1][5][10]. These measures depend on training data as well as a regularization term in an implicit way which make it difficult to see direct effects of a regularization term on generalization error. This naturally leads to our second question.

2) Is it possible to characterize network complexity for a class of networks using only the information given by a regularization term[2]? How to relate the estimated network complexity rigorously with generalization error?

In practice, when a regularization term ($L_{n,N}(w)$) is used to penalize the magnitude of weights, it effectively minimizes the number of hidden units as well even though an additional regularization term $L_N(n)$ is not used. This is due to the fact that some of the hidden units may only operate in the linear region of a sigmoid when their

incoming weights are small and inputs are bounded. Therefore, a $L_{n,N}(w)$ term can effectively act like a $L_N(n)$ term that reduces the effective number of hidden units, and thus result in a degenerate parameter space whose degrees of freedom is fewer than $nd$. This fact was not taken into consideration in the previous work, and as shown later in this work, will lead to a tighter bound on $R_{n,N}$.

In what follows, we will first define the expected network complexity, then use it to bound the generalization error.

## 4    The Expected Network Complexity

For reasons that will become apparent, we choose to define the effective complexity of a feedforward two layer network as the expected number of hidden units $\mathbf{E}K$ ($1 \leq K \leq n$) which are effectively nonlinear, i.e. operating outside the central linear regions of their sigmoid response function $g(z)$. We define the linear region as an interval $|z| < b$ with $b$ a positive constant.

Consider the presynaptic input $z = w'^T x$ to a hidden unit $g(z)$, where $w'$ is the incoming weight vector for the unit. Then the unit is considered to be effectively linear if $|z| < b$ for all $x \in D$. This will happen if $|z'| < b$, where $z' = w'^T x'$ with $x'$ being any vertex of the unit hypercube $D$. This is because $|z| \leq w'^T \hat{x}$, where $\hat{x}$ is the vertex of $D$ whose elements are the $sgn$ functions of the elements of $w'$.

Next, consider network weights as random variables with a distribution $p(w) = A exp(-L_{n,N}(w))$, which corresponds to a Gibbs distribution of a regularization term with a normalizing constant $A$. Consider the vector $x'$ to be a random vector also with equally probable 1's and $-1$'s. Then $|z'| < b$ will be a random event. The probability for this hidden unit to be effectively nonlinear equals to $1 - \Pr(|z| < b)$, which can be completely determined by the distributions of weights $p(w)$ and $x'$ (equally probable). Let $K$ be the number of hidden units which are effectively nonlinear. Then the probability, $\Pr(K = k)$ ($1 \leq k \leq n$), can be determined through a joint probability of $k$ hidden units that are operating beyond the central linear region of sigmoid functions. The expected network complexity, $\mathbf{E}K$, can then be obtained through $\Pr(K = k)$, which is determined by the Gibbs distribution of $L_{N,n}(w)$. The motivation on utilizing such a Gibbs distribution comes from the fact that $R_{k,N}$ is independent of training samples but dependent of a regularization term which corresponds to a prior distribution of weights. Using such a formulation, as will be shown later, the effect of a regularization term on bias and variance can be characterized explicitly.

## 5    A New Bound for The Generalization Error

To develop a tighter bound for the generalization error, we consider subspaces of the weights indexed by different number of effectively nonlinear hidden units: $\Theta_1 \subseteq \Theta_2 ... \subseteq \Theta_n$. For each $\Theta_j$, there are $j$ out of $n$ hidden units which are effectively nonlinear for $1 \leq j \leq n$. Therefore, the index of resolvability $R_{n,N}$ can be expressed as

$$R_{n,N} = \min_{1 \leq k \leq n} R_{k,N}, \qquad (4)$$

where each $R_{k,N} = \min_{w \in \Theta_k} \{ \| f - \hat{f}_n \|^2 + \frac{L_{n,N}(w)}{N} \}$. Next let us consider the number of effectively nonlinear units to be random. Since the minimum is no bigger than the average, we have

$$R_{n,N} \leq \mathbf{E} R_{K,N}, \tag{5}$$

where the expectation is taken over the random variable $K$ utilizing the probability $\Pr(K = k)$. For each $K$, however, the two terms in $R_{K,N}$ can be bounded as

$$\| f - f_{n-K,n} \|^2 \leq O(\| f - f_K \|^2) + O(\| f_{n-K,n} - f_K \|^2), \tag{6}$$

by the triangle inequality, where $f_{n-K,n}$ is the actual network function with $n - K$ hidden units operating in the region bounded by the constant $b$, and $f_K$ is the corresponding network function which treats the $n - K$ units as linear units. In addition, we have

$$L_{n,N}(w) \leq O(\| f_{n-K,n} - f_K \|^2) + O(\frac{Kd}{N} logN), \tag{7}$$

where the first term also results from the triangle inequality, and the second term is obtained by discretizing the degenerate parameter space $\Theta_K$ using similar techniques as in [2][3]. Applying Taylor expansion on the term $\| f_{n-K,n} - f_K \|^2$, we have

$$\| f_{n-K,n} - f_K \|^2 \leq O(b^6(n - K)). \tag{8}$$

Putting Equations (5) (6) (7) and (8) into Equation (1), we have

$$E_g \leq O(\mathbf{E}\frac{1}{K}) + O(\frac{(\mathbf{E}K)d}{N} logN) + O(b^6(n - \mathbf{E}K)) + o(b^6), \tag{9}$$

where $\mathbf{E}K$ is the expected number of hidden units which are effectively nonlinear. If $b \leq O(\frac{1}{n^{\frac{1}{3}}})$, we have

$$E_g \leq O(\mathbf{E}\frac{1}{K}) + O(\frac{(\mathbf{E}K)d}{N} logN). \tag{10}$$

## 6    A Closed Form Expression For a Class of Regularization Terms

For commonly used regularization terms, how can we actually find the probability distribution of the number of (nonlinear) hidden units $\Pr(K = k)$? And how shall we evaluate $\mathbf{E}K$ and $\mathbf{E}\frac{1}{K}$?

As a simple example, we consider a special class of prior distributions for iid weights, i.e, $p(w) = \Pi_i p(w_i)$, where $w_i$ are the elements of $w \in \Theta_n$. This corresponds to a large class of regularization terms which minimize the magnitudes of individual weights independently[7].

Consider each weight as a random variable with zero mean and a common variance $\sigma$. Then for large input dimension $d$, $\frac{1}{\sqrt{d}} z'$ is approximately normal with zero-mean

and variance $\sigma$ by the Central Limit Theorem[3]. Let $q$ denote the probability that a unit is effectively nonlinear. We have

$$q = 2Q(-\frac{b}{\sigma\sqrt{d}}), \tag{11}$$

where $Q(-x) = \frac{1}{\sqrt{2\pi}} \int_{-\infty}^{-x} e^{-\frac{y^2}{2}} dy$. Next consider the probability that $K$ out of $n$ hidden units are nonlinear. Based on our (independence) assumptions on $w'$ and $x'$, $K$ has a binomial distribution

$$\Pr(K = k) = \binom{n}{k} q^k (1 - q)^{n-k}, \tag{12}$$

where $1 \le k \le n$. Then

$$\mathbf{E}K = nq. \tag{13}$$

$$\mathbf{E}\frac{1}{K} = \frac{1}{n} + \Delta, \tag{14}$$

where $\Delta = \sum_{i=1}^{n-1} \frac{1}{i}(1-q)^{n-i} + (1-q)^n$. Then the generalization error $E_g$ satisfies

$$E_g \le O(\frac{1}{n} + \Delta) + O(\frac{nqd}{N} log N) \tag{15}$$

## 7  Application

As an example for applications of the theoretical results, the expected network complexity $\mathbf{E}K$ is estimated for Gaussian mixture model used for time-series prediction (details can be found in [6]) [4].

In general, using only a prior distribution of weights to estimate the network complexity $\mathbf{E}K$ may lead to a less accurate measure on the effective network complexity than incorporating information on training data also. However, if parameters of a regularization term also get optimized during training, as shown in this example, the resulting Gibbs prior distribution of weights may lead to a good estimate of the effective number of hidden units.

Specifically, the corresponding Gibbs distribution $p(w)$ of the weights from the Gaussion mixture is iid, which consists of a linear combination of eight Gaussian distributions. This function results in a skewed distribution with a sharp peak around the zero (see [6]). The mean and variance of the presynaptic inputs $z$ to the hidden units can thus be estimated as 0.02 and 0.04, respectively. The other parameters used are $n = 8$, $d = 12$. $b = 0.6$ is chosen. Then $q \approx 0.4$ is obtained through Equation (11). The effective network complexity is $EK \approx 3$ (or 4). The empirical result[10], which estimates the effective number of hidden units using the dominated eigenvalues at the hidden layer, results in about 3 effective hidden units.

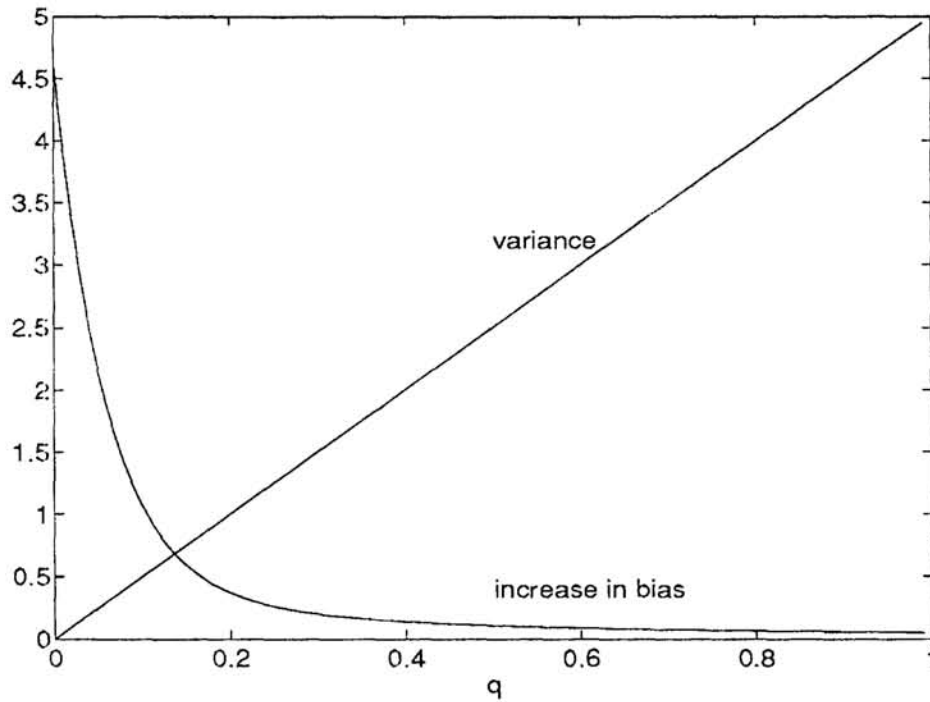

Figure 1: Illustration of an increase $\Delta$ in bias and variance $Bqn$ as a function of $q$. A scaling factor $B = 0.25$ is used for the convenience of the plot. $n = 20$ is chosen.

## 8    Discussions

Is this new bound for the generalization tighter than the old one which takes no account of network-weight-dependent information? If so, what does it tell us?

Compared with the bound in Equation (3), the new bound results in an increase $\Delta$ in approximation error (bias), and $qn$ instead of $n$ as estimation error (variance). These two terms are plotted as functions of $q$ in Figure (1). Since $q$ is a function of $\sigma$ which characterizes how strongly the magnitude of the weights is penalized, the larger the $\sigma$, the less the weights get penalized, the larger the $q$, the more hidden units are likely to be effectively nonlinear, thus the smaller the bias and larger the variance. When $q = 1$, all the hidden units are effectively nonlinear and the new bound reduces to the old one. This shows how a regularization term directly affects bias/variance.

When the estimation error dominates, the bound for the generalization error will be proportional to $nq$ instead of $n$. The value of $nq$, however, depends on the choice of $\sigma$. For small $\sigma$, the new bound can be much tighter than the old one, especially for large networks with $n$ large but $nq$ small. This will provide a practical method to estimate generalization error for large networks as well as an explanation of when and why large networks can generalize well.

How tight the bound really is depends on how well $L_{n,\Lambda}(w)$ is chosen. Let $n_0$ denote the optimal number of (nonlinear) hidden units needed to approximate $f(x)$. If $L_{n,N}(w)$ is chosen so that the corresponding $p(w)$ is almost a delta function at $n_0$, then $\mathbf{E}R_{K,N} \approx R_{n_0,N}$, which gives a very tight bound. Otherwise, if, for instance,

$L_{n,N}(w)$ penalizes network complexity so little that $\mathbf{E}R_{K,N} \approx R_{n,N}$, the bound will be as loose as the original one. It should also be noted that an exact value for the bound cannot be obtained unless some information on the unknown function $f$ itself is available.

For commonly used regularization terms, the expected network complexity can be estimated through a close form expression. Such expected network complexity is shown to be a good estimate for the actual network complexity if a Gibbs prior distribution of weights also gets optimized through training, and is also sharply peaked. More research will be done to evaluate the applicability of the theoretical results.

**Acknowledgement**

The support of National Science Foundation is gratefully acknowledged.

## Footnotes

[1]In the previous work by Barron, the sigmoidal hidden units are $\frac{g_l(z)+1}{2}$. It is easy to show that his results are applicable to the class of $g_l(z)$'s we consider here.

[2]This was posed as an open problem by Solla et.al. [8]

[3]Details will be given in a longer version of the paper in preparation.

[4] Strictly speaking, the theoretical results deal with regularization terms with discrete weights. It can and has been extended to continuous weights by D.F. McCaffrey and A.R. Gallant. Details are beyond the content of this paper.

# References

[1] S. Amari and N. Murata, "Statistical Theory of Learning Curves under Entropic Loss Criterion," *Neural Computation*, 5, 140-153, 1993.

[2] A. Barron, "Approximation and Estimation Bounds for Artificial Neural Networks," *Proc. of The 4th Workshop on Computational Learning Theory*, 243-249, 1991.

[3] W. Feller, *An Introduction to Probability Theory and Its Applications*, New York: John Wiley and Sons, 1968.

[4] S. Geman, E. Bienenstock, and R. Doursat, "Neural Networks and the Bias/Variance Dilemma," *Neural Computation*, 4, 1-58, 1992.

[5] J. Moody, "Generalization, Weight Decay, and Architecture Selection for Nonlinear Learning Systems," *Proc. of Neural Information Processing Systems*, 1991.

[6] S.J. Nowlan, and G.E. Hinton, "Simplifying Neural Networks by Soft Weight Sharing," *Neural computation*, 4, 473-493(1992).

[7] R. Reed, "Pruning Algorithms-A Survey," *IEEE Trans. Neural Networks* Vol. 4, 740-747, (1993).

[8] S. Solla, "The Emergence of Generalization Ability in Learning," Presented at *NIPS92*.

[9] V. Vapnik, "Estimation of Dependences Based on Empirical Data," *Springer-Verlag*, New York, 1982.

[10] A.S . Weigend and D.E . Rumelhart, "The Effective Dimension of the Space of Hidden Units," *Proc. of International Joint Conference on Neural Networks*, 1992.
